# Efficient Monte Carlo Counterfactual Regret Minimization in Games with Many Player Actions

**Richard Gibson, Neil Burch, Marc Lanctot, and Duane Szafron**
Department of Computing Science, University of Alberta
Edmonton, Alberta, T6G 2E8, Canada
{rggibson | nburch | lanctot | dszafron}@ualberta.ca

## Abstract

Counterfactual Regret Minimization (CFR) is a popular, iterative algorithm for computing strategies in extensive-form games. The Monte Carlo CFR (MCCFR) variants reduce the per iteration time cost of CFR by traversing a smaller, sampled portion of the tree. The previous most effective instances of MCCFR can still be very slow in games with many player actions since they sample every action for a given player. In this paper, we present a new MCCFR algorithm, Average Strategy Sampling (AS), that samples a subset of the player's actions according to the player's average strategy. Our new algorithm is inspired by a new, tighter bound on the number of iterations required by CFR to converge to a given solution quality. In addition, we prove a similar, tighter bound for AS and other popular MCCFR variants. Finally, we validate our work by demonstrating that AS converges faster than previous MCCFR algorithms in both no-limit poker and Bluff.

## 1   Introduction

An extensive-form game is a common formalism used to model sequential decision making problems containing multiple agents, imperfect information, and chance events. A typical solution concept in games is a Nash equilibrium profile. Counterfactual Regret Minimization (CFR) [12] is an iterative algorithm that, in 2-player zero-sum extensive-form games, converges to a Nash equilibrium. Other techniques for computing Nash equilibria of 2-player zero-sum games include linear programming [8] and the Excessive Gap Technique [6]. Theoretical results indicate that for a fixed solution quality, CFR takes a number of iterations at most quadratic in the size of the game [12, Theorem 4]. Thus, as we consider larger games, more iterations are required to obtain a fixed solution quality. Nonetheless, CFR's versatility and memory efficiency make it a popular choice.

Monte Carlo CFR (MCCFR) [9] can be used to reduce the traversal time per iteration by considering only a sampled portion of the game tree. For example, Chance Sampling (CS) [12] is an instance of MCCFR that only traverses the portion of the game tree corresponding to a single, sampled sequence of chance's actions. However, in games where a player has many possible actions, such as no-limit poker, iterations of CS are still very time consuming. This is because CS considers all possible player actions, even if many actions are poor or only factor little into the algorithm's computation.

Our main contribution in this paper is a new MCCFR algorithm that samples player actions and is suitable for games involving many player choices. Firstly, we provide tighter theoretical bounds on the number of iterations required by CFR and previous MCCFR algorithms to reach a fixed solution quality. Secondly, we use these new bounds to propel our new MCCFR sampling algorithm. By using a player's *average strategy* to sample actions, convergence time is significantly reduced in large games with many player actions. We prove convergence and show that our new algorithm approaches equilibrium faster than previous sampling schemes in both no-limit poker and Bluff.

## 2 Background

A finite extensive game contains a game tree with nodes corresponding to **histories** of actions $h \in H$ and edges corresponding to **actions** $a \in A(h)$ available to **player** $P(h) \in N \cup \{c\}$ (where $N$ is the set of players and $c$ denotes **chance**). When $P(h) = c$, $\sigma_c(h, a)$ is the (fixed) probability of chance generating action $a$ at $h$. Each **terminal history** $z \in Z$ has associated **utilities** $u_i(z)$ for each player $i$. We define $\Delta_i = \max_{z, z' \in Z} u_i(z) - u_i(z')$ to be the range of utilities for player $i$. Non-terminal histories are partitioned into **information sets** $I \in \mathcal{I}_i$ representing the different game states that player $i$ cannot distinguish between. For example, in poker, player $i$ does not see the private cards dealt to the opponents, and thus all histories differing only in the private cards of the opponents are in the same information set for player $i$. The action sets $A(h)$ must be identical for all $h \in I$, and we denote this set by $A(I)$. We define $|A_i| = \max_{I \in \mathcal{I}_i} |A(I)|$ to be the maximum number of actions available to player $i$ at any information set. We assume **perfect recall** that guarantees players always remember information that was revealed to them and the order in which it was revealed.

A **(behavioral) strategy for player $i$**, $\sigma_i \in \Sigma_i$, is a function that maps each information set $I \in \mathcal{I}_i$ to a probability distribution over $A(I)$. A **strategy profile** is a vector of strategies $\sigma = (\sigma_1, ..., \sigma_{|N|}) \in \Sigma$, one for each player. Let $u_i(\sigma)$ denote the expected utility for player $i$, given that all players play according to $\sigma$. We let $\sigma_{-i}$ refer to the strategies in $\sigma$ excluding $\sigma_i$. Let $\pi^\sigma(h)$ be the probability of history $h$ occurring if all players choose actions according to $\sigma$. We can decompose $\pi^\sigma(h) = \prod_{i \in N \cup \{c\}} \pi_i^\sigma(h)$, where $\pi_i^\sigma(h)$ is the contribution to this probability from player $i$ when playing according to $\sigma_i$ (or from chance when $i = c$). Let $\pi_{-i}^\sigma(h)$ be the product of all players' contributions (including chance) except that of player $i$. Let $\pi^\sigma(h, h')$ be the probability of history $h'$ occurring after $h$, given $h$ has occurred. Furthermore, for $I \in \mathcal{I}_i$, the probability of player $i$ playing to reach $I$ is $\pi_i^\sigma(I) = \pi_i^\sigma(h)$ for any $h \in I$, which is well-defined due to perfect recall.

A **best response** to $\sigma_{-i}$ is a strategy that maximizes player $i$'s expected payoff against $\sigma_{-i}$. The **best response value** for player $i$ is the value of that strategy, $b_i(\sigma_{-i}) = \max_{\sigma_i' \in \Sigma_i} u_i(\sigma_i', \sigma_{-i})$. A strategy profile $\sigma$ is an **$\epsilon$-Nash equilibrium** if no player can unilaterally deviate from $\sigma$ and gain more than $\epsilon$; *i.e.*, $u_i(\sigma) + \epsilon \geq b_i(\sigma_{-i})$ for all $i \in N$. A game is **two-player zero-sum** if $N = \{1, 2\}$ and $u_1(z) = -u_2(z)$ for all $z \in Z$. In this case, the **exploitability** of $\sigma$, $e(\sigma) = (b_1(\sigma_2) + b_2(\sigma_1))/2$, measures how much $\sigma$ loses to a worst case opponent when players alternate positions. A 0-Nash equilibrium (or simply a **Nash equilibrium**) has zero exploitability.

**Counterfactual Regret Minimization (CFR)** [12] is an iterative algorithm that, for two-player zero sum games, computes an $\epsilon$-Nash equilibrium profile with $\epsilon \to 0$. CFR has also been shown to work well in games with more than two players [1, 3]. On each iteration $t$, the base algorithm, "vanilla" CFR, traverses the entire game tree once per player, computing the expected utility for player $i$ at each information set $I \in \mathcal{I}_i$ under the current profile $\sigma^t$, assuming player $i$ plays to reach $I$. This expectation is the **counterfactual value** for player $i$, $v_i(I, \sigma) = \sum_{z \in Z_I} u_i(z) \pi_{-i}^\sigma(z[I]) \pi^\sigma(z[I], z)$, where $Z_I$ is the set of terminal histories passing through $I$ and $z[I]$ is that history along $z$ contained in $I$. For each action $a \in A(I)$, these values determine the **counterfactual regret** at iteration $t$,

$$r_i^t(I, a) = v_i(I, \sigma_{(I \to a)}^t) - v_i(I, \sigma^t),$$

where $\sigma_{(I \to a)}$ is the profile $\sigma$ except that at $I$, action $a$ is always taken. The regret $r_i^t(I, a)$ measures how much player $i$ would rather play action $a$ at $I$ than play $\sigma^t$. These regrets are accumulated to obtain the **cumulative counterfactual regret**, $R_i^T(I, a) = \sum_{t=1}^T r_i^t(I, a)$, and are used to update the current strategy profile via regret matching [5, 12],

$$\sigma^{T+1}(I, a) = \frac{R_i^{T,+}(I, a)}{\sum_{b \in A(I)} R_i^{T,+}(I, b)}, \tag{1}$$

where $x^+ = \max\{x, 0\}$ and actions are chosen uniformly at random when the denominator is zero. It is well-known that in a two-player zero-sum game, if both players' **average (external) regret**,

$$\frac{R_i^T}{T} = \max_{\sigma_i' \in \Sigma_i} \frac{1}{T} \sum_{t=1}^T \left( u_i(\sigma_i', \sigma_{-i}^t) - u_i(\sigma_i^t, \sigma_{-i}^t) \right),$$

is at most $\epsilon/2$, then the **average profile** $\bar{\sigma}^T$ is an $\epsilon$-Nash equilibrium. During computation, CFR stores a **cumulative profile** $s_i^T(I, a) = \sum_{t=1}^T \pi_i^{\sigma^t}(I) \sigma_i^t(I, a)$ and outputs the average profile

$\bar{\sigma}_i^T(I,a) = s_i^T(I,a)/\sum_{b \in A(I)} s_i^T(I,b)$. The original CFR analysis shows that player $i$'s regret is bounded by the sum of the positive parts of the cumulative counterfactual regrets $R_i^{T,+}(I,a)$:

**Theorem 1 (Zinkevich *et al.* [12])**

$$R_i^T \le \sum_{I \in \mathcal{I}} \max_{a \in A(I)} R_i^{T,+}(I,a).$$

Regret matching minimizes the average of the cumulative counterfactual regrets, and so player $i$'s average regret is also minimized by Theorem 1. For each player $i$, let $\mathcal{B}_i$ be the partition of $\mathcal{I}_i$ such that two information sets $I, I'$ are in the same part $B \in \mathcal{B}_i$ if and only if player $i$'s sequence of actions leading to $I$ is the same as the sequence of actions leading to $I'$. $\mathcal{B}_i$ is well-defined due to perfect recall. Next, define the **$M$-value of the game to player $i$** to be $M_i = \sum_{B \in \mathcal{B}_i} \sqrt{|B|}$. The best known bound on player $i$'s average regret is:

**Theorem 2 (Lanctot *et al.* [9])** *When using vanilla CFR, average regret is bounded by*

$$\frac{R_i^T}{T} \le \frac{\Delta_i M_i \sqrt{|A_i|}}{\sqrt{T}}.$$

We prove a tighter bound in Section 3. For large games, CFR's full game tree traversals can be very expensive. Alternatively, one can traverse a smaller, sampled portion of the tree on each iteration using **Monte Carlo CFR (MCCFR)** [9]. Let $\mathcal{Q} = \{Q_1, ..., Q_K\}$ be a set of subsets, or **blocks**, of the terminal histories $Z$ such that the union of $\mathcal{Q}$ spans $Z$. For example, **Chance Sampling (CS)** [12] is an instance of MCCFR that partitions $Z$ into blocks such that two histories are in the same block if and only if no two chance actions differ. On each iteration, a block $Q_j$ is sampled with probability $q_j$, where $\sum_{k=1}^{K} q_k = 1$. In CS, we generate a block by sampling a single action $a$ at each history $h \in H$ with $P(h) = c$ according to its likelihood of occurring, $\sigma_c(h,a)$. In general, the **sampled counterfactual value** for player $i$ is

$$\tilde{v}_i(I,\sigma) = \sum_{z \in Z_I \cap Q_j} u_i(z) \pi_{-i}^{\sigma}(z[I]) \pi^{\sigma}(z[I], z)/q(z),$$

where $q(z) = \sum_{k:z \in Q_k} q_k$ is the probability that $z$ was sampled. For example, in CS, $q(z) = \pi_c^{\sigma}(z)$. Define the **sampled counterfactual regret** for action $a$ at $I$ to be $\tilde{r}_i^t(I,a) = \tilde{v}_i(I, \sigma_{(I \to a)}^t) - \tilde{v}_i(I, \sigma^t)$. Strategies are then generated by applying regret matching to $\tilde{R}_i^T(I,a) = \sum_{t=1}^{T} \tilde{r}_i^t(I,a)$.

CS has been shown to significantly reduce computing time in poker games [11, Appendix A.5.2]. Other instances of MCCFR include **External Sampling (ES)** and **Outcome Sampling (OS)** [9]. ES takes CS one step further by considering only a single action for not only chance, but also for the opponents, where opponent actions are sampled according to the current profile $\sigma_{-i}^t$. OS is the most extreme version of MCCFR that samples a single action at every history, walking just a single trajectory through the tree on each traversal ($Q_j = \{z\}$). ES and OS converge to equilibrium faster than vanilla CFR in a number of different domains [9, Figure 1].

ES and OS yield a probabilistic bound on the average regret, and thus provide a probabilistic guarantee that $\bar{\sigma}^T$ converges to a Nash equilibrium. Since both algorithms generate blocks by sampling actions independently, we can decompose $q(z) = \prod_{i \in N \cup \{c\}} q_i(z)$ so that $q_i(z)$ is the probability contributed to $q(z)$ by sampling player $i$'s actions.

**Theorem 3 (Lanctot *et al.* [9])** [1] *Let $X$ be one of ES or OS (assuming OS also samples opponent actions according to $\sigma_{-i}$), let $p \in (0,1]$, and let $\delta = \min_{z \in Z} q_i(z) > 0$ over all $1 \le t \le T$. When using $X$, with probability $1 - p$, average regret is bounded by*

$$\frac{R_i^T}{T} \le \left( M_i + \frac{\sqrt{2|\mathcal{I}_i||\mathcal{B}_i|}}{\sqrt{p}} \right) \left( \frac{1}{\delta} \right) \frac{\Delta_i \sqrt{|A_i|}}{\sqrt{T}}.$$

## 3 New CFR Bounds

While Zinkevich *et al.* [12] bound a player's regret by a sum of cumulative counterfactual regrets (Theorem 1), we can actually *equate* a player's regret to a weighted sum of counterfactual regrets. For a strategy $\sigma_i \in \Sigma_i$ and an information set $I \in \mathcal{I}_i$, define $R_i^T(I, \sigma_i) = \sum_{a \in A(I)} \sigma_i(I, a) R_i^T(I, a)$. In addition, let $\sigma_i^* \in \Sigma_i$ be a player $i$ strategy such that $\sigma_i^* = \arg\max_{\sigma_i' \in \Sigma_i} \sum_{t=1}^{T} u_i(\sigma_i', \sigma_{-i}^t)$. Note that in a two-player game, $\sum_{t=1}^{T} u_i(\sigma_i^*, \sigma_{-i}^t) = T u_i(\sigma_i^*, \bar{\sigma}_{-i}^T)$, and thus $\sigma_i^*$ is a best response to the opponent's average strategy after $T$ iterations.

**Theorem 4**

$$R_i^T = \sum_{I \in \mathcal{I}_i} \pi^{\sigma^*}(I) R_i^T(I, \sigma_i^*).$$

All proofs in this paper are provided in full as supplementary material. Theorem 4 leads to a tighter bound on the average regret when using CFR. For a strategy $\sigma_i \in \Sigma_i$, define the **$M$-value of $\sigma_i$** to be $M_i(\sigma_i) = \sum_{B \in \mathcal{B}_i} \pi_i^\sigma(B) \sqrt{|B|}$, where $\pi_i^\sigma(B) = \max_{I \in B} \pi_i^\sigma(I)$. Clearly, $M_i(\sigma_i) \leq M_i$ for all $\sigma_i \in \Sigma_i$ since $\pi_i^\sigma(B) \leq 1$. For vanilla CFR, we can simply replace $M_i$ in Theorem 2 with $M_i(\sigma_i^*)$:

**Theorem 5** *When using vanilla CFR, average regret is bounded by*

$$\frac{R_i^T}{T} \leq \frac{\Delta_i M_i(\sigma_i^*) \sqrt{|A_i|}}{\sqrt{T}}.$$

For MCCFR, we can show a similar improvement to Theorem 3. Our proof includes a bound for CS that appears to have been omitted in previous work. Details are in the supplementary material.

**Theorem 6** *Let $X$ be one of CS, ES, or OS (assuming OS samples opponent actions according to $\sigma_{-i}$), let $p \in (0, 1]$, and let $\delta = \min_{z \in Z} q_i(z) > 0$ over all $1 \leq t \leq T$. When using $X$, with probability $1 - p$, average regret is bounded by*

$$\frac{R_i^T}{T} \leq \left( M_i(\sigma_i^*) + \frac{\sqrt{2|\mathcal{I}_i||\mathcal{B}_i|}}{\sqrt{p}} \right) \left( \frac{1}{\delta} \right) \frac{\Delta_i \sqrt{|A_i|}}{\sqrt{T}}.$$

Theorem 4 states that player $i$'s regret is equal to the weighted sum of player $i$'s counterfactual regrets at each $I \in \mathcal{I}_i$, where the weights are equal to player $i$'s probability of reaching $I$ under $\sigma_i^*$. Since our goal is to minimize average regret, this means that we only need to minimize the average cumulative counterfactual regret at each $I \in \mathcal{I}_i$ that $\sigma_i^*$ plays to reach. Therefore, when using MCCFR, we may want to sample more often those information sets that $\sigma_i^*$ plays to reach, and less often those information sets that $\sigma_i^*$ avoids. This inspires our new MCCFR sampling algorithm.

## 4 Average Strategy Sampling

Leveraging the theory developed in the previous section, we now introduce a new MCCFR sampling algorithm that can minimize average regret at a faster rate than CS, ES, and OS. As we just described, we want our algorithm to sample more often the information sets that $\sigma_i^*$ plays to reach. Unfortunately, we do not have the exact strategy $\sigma_i^*$ on hand. Recall that in a two-player game, $\sigma_i^*$ is a best response to the opponent's average strategy, $\bar{\sigma}_{-i}^T$. However, for two-player zero-sum games, we do know that the average profile $\bar{\sigma}^T$ converges to a Nash equilibrium. This means that player $i$'s average strategy, $\bar{\sigma}_i^T$, converges to a best response of $\bar{\sigma}_{-i}^T$. While the average strategy is not an exact best response, it can be used as a heuristic to guide sampling within MCCFR. Our new sampling algorithm, **Average Strategy Sampling (AS)**, selects actions for player $i$ according to the cumulative profile and three predefined parameters. AS can be seen as a sampling scheme between OS and ES where a subset of player $i$'s actions are sampled at each information set $I$, as opposed to sampling one action (OS) or sampling every action (ES). Given the cumulative profile $s_i^T(I, \cdot)$ on iteration $T$, an exploration parameter $\epsilon \in (0, 1]$, a threshold parameter $\tau \in [1, \infty)$, and a bonus parameter $\beta \in [0, \infty)$, each of player $i$'s actions $a \in A(I)$ are sampled independently with probability

$$\rho(I, a) = \max \left\{ \epsilon, \frac{\beta + \tau s_i^T(I, a)}{\beta + \sum_{b \in A(I)} s_i^T(I, b)} \right\}, \tag{2}$$

---

**Algorithm 1** Average Strategy Sampling (Two-player version)

---

1: **Require:** Parameters $\epsilon, \tau, \beta$
2: Initialize regret and cumulative profile: $\forall I, a : r(I,a) \leftarrow 0, s(I,a) \leftarrow 0$
3:
4: WalkTree(history $h$, player $i$, sample prob $q$):
5:      **if** $h \in Z$ **then return** $u_i(h)/q$ **end if**
6:      **if** $h \in P(c)$ **then** Sample action $a \sim \sigma_c(h, \cdot)$, **return** WalkTree($ha$, $i$, $q$) **end if**
7:      $I \leftarrow$ Information set containing $h$ , $\sigma(I, \cdot) \leftarrow$ RegretMatching($r(I, \cdot)$)
8:      **if** $h \notin P(i)$ **then**
9:          **for** $a \in A(I)$ **do** $s(I,a) \leftarrow s(I,a) + (\sigma(I,a)/q)$ **end for**
10:         Sample action $a \sim \sigma(I, \cdot)$, **return** WalkTree($ha$, $i$, $q$)
11:      **end if**
12:      **for** $a \in A(I)$ **do**
13:          $\rho \leftarrow \max \left\{ \epsilon, \frac{\beta + \tau s(I,a)}{\beta + \sum_{b \in A(I)} s(I,b)} \right\}, \tilde{v}(a) \leftarrow 0$
14:          **if** Random$(0,1) < \rho$ **then** $\tilde{v}(a) \leftarrow$ WalkTree($ha$, $i$, $q \cdot \min\{1, \rho\}$) **end if**
15:      **end for**
16:      **for** $a \in A(I)$ **do** $r(I,a) \leftarrow r(I,a) + \tilde{v}(a) - \sum_{a \in A(I)} \sigma(I,a)\tilde{v}(a)$ **end for**
17:      **return** $\sum_{a \in A(I)} \sigma(I,a)\tilde{v}(a)$

---

or with probability 1 if either $\rho(I,a) > 1$ or $\beta + \sum_{b \in A(I)} s_i^T(I,b) = 0$. As in ES, at opponent and chance nodes, a single action is sampled on-policy according to the current opponent profile $\sigma_{-i}^T$ and the fixed chance probabilities $\sigma_c$ respectively.

If $\tau = 1$ and $\beta = 0$, then $\rho(I,a)$ is equal to the probability that the average strategy $\bar{\sigma}_i^T = s_i^T(I,a)/\sum_{b \in A(I)} s_i^T(I,b)$ plays $a$ at $I$, except that each action is sampled with probability at least $\epsilon$. For choices greater than 1, $\tau$ acts as a threshold so that any action taken with probability at least $1/\tau$ by the average strategy is always sampled by AS. Furthermore, $\beta$'s purpose is to increase the rate of exploration during early AS iterations. When $\beta > 0$, we effectively add $\beta$ as a bonus to the cumulative value $s_i^T(I,a)$ before normalizing. Since player $i$'s average strategy $\bar{\sigma}_i^T$ is not a good approximation of $\sigma_i^*$ for small $T$, we include $\beta$ to avoid making ill-informed choices early-on. As the cumulative profile $s_i^T(I, \cdot)$ grows over time, $\beta$ eventually becomes negligible. In Section 5, we present a set of values for $\epsilon$, $\tau$, and $\beta$ that work well across all of our test games.

Pseudocode for a two-player version of AS is presented in Algorithm 1. In Algorithm 1, the recursive function WalkTree considers four different cases. Firstly, if we have reached a terminal node, we return the utility scaled by $1/q$ (line 5), where $q = q_i(z)$ is the probability of sampling $z$ contributed from player $i$'s actions. Secondly, when at a chance node, we sample a single action according to $\sigma_c$ and recurse down that action (line 6). Thirdly, at an opponent's choice node (lines 8 to 11), we again sample a single action and recurse, this time according to the opponent's current strategy obtained via regret matching (equation (1)). At opponent nodes, we also update the cumulative profile (line 9) for reasons that we describe in a previous paper [2, Algorithm 1]. For games with more than two players, a second tree walk is required and we omit these details.

The final case in Algorithm 1 handles choice nodes for player $i$ (lines 7 to 17). For each action $a$, we compute the probability $\rho$ of sampling $a$ and stochastically decide whether to sample $a$ or not, where Random(0,1) returns a random real number in $[0,1)$. If we do sample $a$, then we recurse to obtain the sampled counterfactual value $\tilde{v}(a) = \tilde{v}_i(I, \sigma_{(I \to a)}^t)$ (line 14). Finally, we update the regrets at $I$ (line 16) and return the sampled counterfactual value at $I$, $\sum_{a \in A(I)} \sigma(I,a)\tilde{v}(a) = \tilde{v}_i(I, \sigma^t)$.

Repeatedly running WalkTree($\emptyset$, $i$, 1) $\forall i \in N$ provides a probabilistic guarantee that all players' average regret will be minimized. In the supplementary material, we prove that AS exhibits the same regret bound as CS, ES, and OS provided in Theorem 6. Note that $\delta$ in Theorem 6 is guaranteed to be positive for AS by the inclusion of $\epsilon$ in equation (2). However, for CS and ES, $\delta = 1$ since all of player $i$'s actions are sampled, whereas $\delta \leq 1$ for OS and AS. While this suggests that fewer iterations of CS or ES are required to achieve the same regret bound compared to OS and AS, iterations for OS and AS are faster as they traverse less of the game tree. Just as CS, ES, and OS

have been shown to benefit from this trade-off over vanilla CFR, we will show that in practice, AS can likewise benefit over CS and ES and that AS is a better choice than OS.

## 5   Experiments

In this section, we compare the convergence rates of AS to those of CS, ES, and OS. While AS can be applied to any extensive game, the aim of AS is to provide faster convergence rates in games involving many player actions. Thus, we consider two domains, no-limit poker and Bluff, where we can easily scale the number of actions available to the players.

**No-limit poker.** The two-player poker game we consider here, which we call **2-NL Hold'em($k$)**, is inspired by no-limit Texas Hold'em. 2-NL Hold'em($k$) is played over two betting rounds. Each player starts with a **stack** of $k$ chips. To begin play, the player denoted as the dealer posts a **small blind** of one chip and the other player posts a **big blind** of two chips. Each player is then dealt two private cards from a standard 52-card deck and the first betting round begins. During each betting round, players can either **fold** (forfeit the game), **call** (match the previous bet), or **raise** by any number of chips in their remaining stack (increase the previous bet), as long as the raise is at least as big as the previous bet. After the first betting round, three public community cards are revealed (the **flop**) and a second and final betting round begins. If a player has no more chips left after a call or a raise, that player is said to be **all-in**. At the end of the second betting round, if neither player folded, then the player with the highest ranked five-card poker hand wins all of the chips played. Note that the number of player actions in 2-NL Hold'em($k$) at one information set is at most the starting stack size, $k$. Increasing $k$ adds more betting options and allows for more actions before being all-in.

**Bluff.** Bluff($D_1, D_2$) [7], also known as Liar's Dice, Perduo, and Dudo, is a two-player dice-bidding game played with six-sided dice over a number of rounds. Each player $i$ starts with $D_i$ dice. In each round, players roll their dice and look at the result without showing their opponent. Then, players alternate by bidding a quantity $q$ of a face value $f$ of all dice in play until one player claims that the other is bluffing (*i.e.*, claims that the bid does not hold). To place a new bid, a player must increase $q$ or $f$ of the current bid. A face value of six is considered "wild" and counts as any other face value. The player calling bluff wins the round if the opponent's last bid is incorrect, and loses otherwise. The losing player removes one of their dice from the game and a new round begins. Once a player has no more dice left, that player loses the game and receives a utility of $-1$, while the winning player earns $+1$ utility. The maximum number of player actions at an information set is $6(D_1 + D_2) + 1$ as increasing $D_i$ allows both players to bid higher quantities $q$.

**Preliminary tests.** Before comparing AS to CS, ES, and OS, we first run some preliminary experiments to find a good set of parameter values for $\epsilon$, $\tau$, and $\beta$ to use with AS. All of our preliminary experiments are in two-player 2-NL Hold'em($k$). In poker, a common approach is to create an abstract game by merging similar card dealings together into a single chance action or "bucket" [4]. To keep the size of our games manageable, we employ a five-bucket abstraction that reduces the branching factor at each chance node down to five, where dealings are grouped according to expected hand strength squared as described by Zinkevich *et al.* [12].

Firstly, we fix $\tau = 1000$ and test different values for $\epsilon$ and $\beta$ in 2-NL Hold'em(30). Recall that $\tau = 1000$ implies actions taken by the average strategy with probability at least 0.001 are always sampled by AS. Figure 1a shows the exploitability in the five-bucket abstract game, measured in milli-big-blinds per game (mbb/g), of the profile produced by AS after $10^{12}$ nodes visited. Recall that lower exploitability implies a closer approximation to equilibrium. Each data point is averaged over five runs of AS. The $\epsilon = 0.05$ and $\beta = 10^5$ or $10^6$ profiles are the least exploitable profiles within statistical noise (not shown).

Next, we fix $\epsilon = 0.05$ and $\beta = 10^6$ and test different values for $\tau$. Figure 1b shows the abstract game exploitability over the number of nodes visited by AS in 2-NL Hold'em(30), where again each data point is averaged over five runs. Here, the least exploitable strategies after $10^{12}$ nodes visited are obtained with $\tau = 100$ and $\tau = 1000$ (again within statistical noise). Similar results to Figure 1b hold in 2-NL Hold'em(40) and are not shown. Throughout the remainder of our experiments, we use the fixed set of parameters $\epsilon = 0.05$, $\beta = 10^6$, and $\tau = 1000$ for AS.

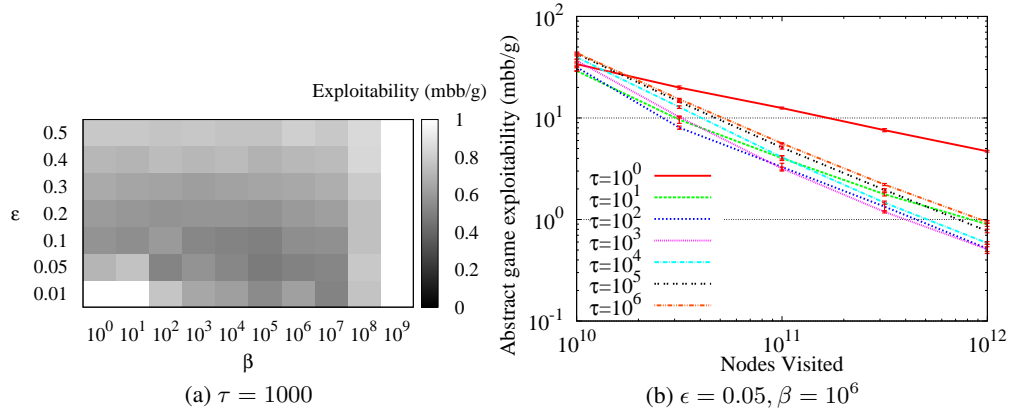

(a) $\tau = 1000$                          (b) $\epsilon = 0.05, \beta = 10^6$

Figure 1: **(a)** Abstract game exploitability of AS profiles for $\tau = 1000$ after $10^{12}$ nodes visited in 2-NL Hold'em(30). **(b)** Log-log plot of abstract game exploitability over the number of nodes visited by AS with $\epsilon = 0.05$ and $\beta = 10^6$ in 2-NL Hold'em(30). For both figures, units are in milli-big-blinds per hand (mbb/g) and data points are averaged over five runs with different random seeds. Error bars in (b) indicate 95% confidence intervals.

**Main results.** We now compare AS to CS, ES, and OS in both 2-NL Hold'em($k$) and Bluff($D_1, D_2$). Similar to Lanctot *et al.* [9], our OS implementation is $\epsilon$-greedy so that the current player $i$ samples a single action at random with probability $\epsilon = 0.5$, and otherwise samples a single action according to the current strategy $\sigma_i$.

Firstly, we consider two-player 2-NL Hold'em($k$) with starting stacks of $k = 20, 22, 24, ..., 38,$ and 40 chips, for a total of eleven different 2-NL Hold'em($k$) games. Again, we apply the same five-bucket card abstraction as before to keep the games reasonably sized. For each game, we ran each of CS, ES, OS, and AS five times, measured the abstract game exploitability at a number of checkpoints, and averaged the results. Figure 2a displays the results for 2-NL Hold'em(36), a game with approximately 68 million information sets and 5 billion histories (nodes). Here, AS achieved an improvement of 54% over ES at the final data points. In addition, Figure 2b shows the average exploitability in each of the eleven games after approximately $3.16 \times 10^{12}$ nodes visited by CS, ES, and AS. OS performed much worse and is not shown. Since one can lose more as the starting stacks are increased (*i.e.*, $\Delta_i$ becomes larger), we "normalized" exploitability across each game by dividing the units on the y-axis by $k$. While there is little difference between the algorithms for the smaller 20 and 22 chip games, we see a significant benefit to using AS over CS and ES for the larger games that contain many player actions. For the most part, the margins between AS, CS, and ES increase with the game size.

Figure 3 displays similar results for Bluff(1, 1) and Bluff(2, 1), which contain over 24 thousand and 3.5 million information sets, and 294 thousand and 66 million histories (nodes) respectively. Again, AS converged faster than CS, ES, and OS in both Bluff games tested. Note that the same choices of parameters ($\epsilon = 0.05, \beta = 10^6, \tau = 1000$) that worked well in 2-NL Hold'em(30) also worked well in other 2-NL Hold'em($k$) games and in Bluff($D_1, D_2$).

## 6 Conclusion

This work has established a number of improvements for computing strategies in extensive-form games with CFR, both theoretically and empirically. We have provided new, tighter bounds on the average regret when using vanilla CFR or one of several different MCCFR sampling algorithms. These bounds were derived by showing that a player's regret is equal to a weighted sum of the player's cumulative counterfactual regrets (Theorem 4), where the weights are given by a best response to the opponents' previous sequence of strategies. We then used this bound as inspiration for our new MCCFR algorithm, AS. By sampling a subset of a player's actions, AS can provide faster

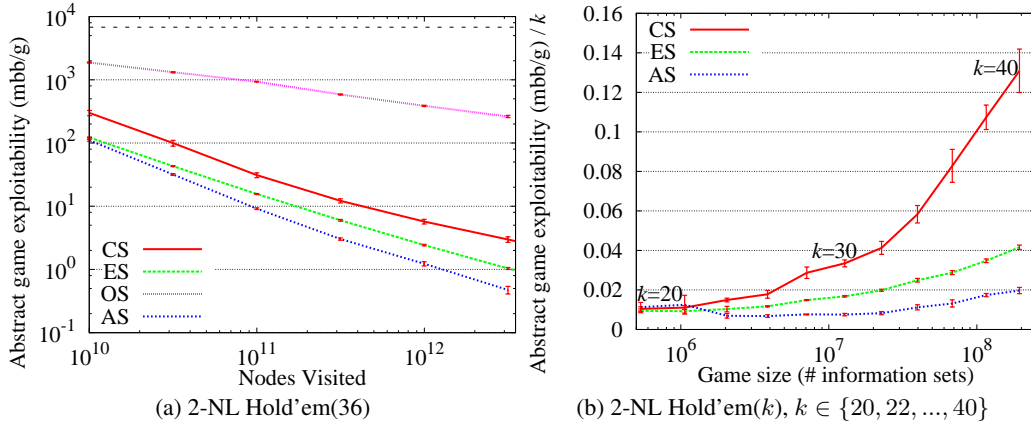

(a) 2-NL Hold'em(36)  (b) 2-NL Hold'em($k$), $k \in \{20, 22, ..., 40\}$

Figure 2: **(a)** Log-log plot of abstract game exploitability over the number of nodes visited by CS, ES, OS, and AS in 2-NL Hold'em(36). The initial uniform random profile is exploitable for 6793 mbb/g, as indicated by the black dashed line. **(b)** Abstract game exploitability after approximately $3.16 \times 10^{12}$ nodes visited over the game size for 2-NL Hold'em($k$) with even-sized starting stacks $k$ between 20 and 40 chips. For both graphs, units are in milli-big-blinds per hand (mbb/g) and data points are averaged over five runs with different random seeds. Error bars indicate 95% confidence intervals. For (b), units on the y-axis are normalized by dividing by the starting chip stacks.

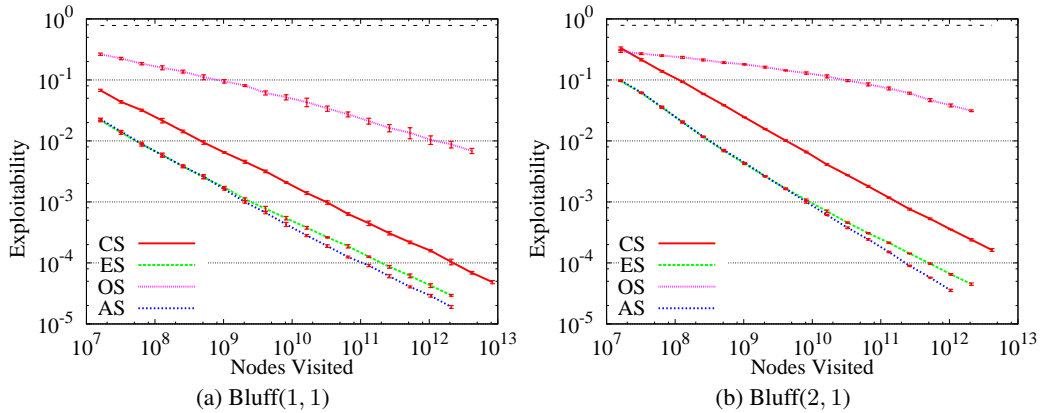

(a) Bluff(1, 1)  (b) Bluff(2, 1)

Figure 3: Log-log plots of exploitability over number of nodes visited by CS, ES, OS, and AS in Bluff(1, 1) and Bluff(2, 1). The initial uniform random profile is exploitable for 0.780 and 0.784 in Bluff(1, 1) and Bluff(2, 1) respectively, as indicated by the black dashed lines. Data points are averaged over five runs with different random seeds and error bars indicate 95% confidence intervals.

convergence rates in games containing many player actions. AS converged faster than previous MC-CFR algorithms in all of our test games. For future work, we would like to apply AS to games with many player actions and with more than two players. All of our theory still applies, except that player $i$'s average strategy is no longer guaranteed to converge to $\sigma_i^*$. Nonetheless, AS may still find strong strategies faster than CS and ES when it is too expensive to sample all of a player's actions.

## Acknowledgments

We thank the members of the Computer Poker Research Group at the University of Alberta for helpful conversations pertaining to this work. This research was supported by NSERC, Alberta Innovates – Technology Futures, and computing resources provided by WestGrid and Compute Canada.

## Footnotes

[1]The bound presented by Lanctot *et al.* appears slightly different, but the last step of their proof mistakenly used $M_i \ge \sqrt{|\mathcal{I}_i||\mathcal{B}_i|}$, which is actually incorrect. The bound we present here is correct.

# References

[1] Nick Abou Risk and Duane Szafron. Using counterfactual regret minimization to create competitive multiplayer poker agents. In *Ninth International Conference on Autonomous Agents and Multiagent Systems (AAMAS)*, pages 159–166, 2010.

[2] Richard Gibson, Marc Lanctot, Neil Burch, Duane Szafron, and Michael Bowling. Generalized sampling and variance in counterfactual regret minimization. In *Twenty-Sixth Conference on Artificial Intelligence (AAAI)*, pages 1355–1361, 2012.

[3] Richard Gibson and Duane Szafron. On strategy stitching in large extensive form multiplayer games. In *Advances in Neural Information Processing Systems 24 (NIPS)*, pages 100–108, 2011.

[4] Andrew Gilpin and Tuomas Sandholm. A competitive Texas Hold'em poker player via automated abstraction and real-time equilibrium computation. In *Twenty-First Conference on Artificial Intelligence (AAAI)*, pages 1007–1013, 2006.

[5] Sergiu Hart and Andreu Mas-Colell. A simple adaptive procedure leading to correlated equilibrium. *Econometrica*, **68**:1127–1150, 2000.

[6] Samid Hoda, Andrew Gilpin, Javier Peña, and Tuomas Sandholm. Smoothing techniques for computing Nash equilibria of sequential games. *Mathematics of Operations Research*, **35**(2):494–512, 2010.

[7] Reiner Knizia. *Dice Games Properly Explained*. Blue Terrier Press, 2010.

[8] Daphne Koller, Nimrod Megiddo, and Bernhard von Stengel. Fast algorithms for finding randomized strategies in game trees. In *Annual ACM Symposium on Theory of Computing (STOC'94)*, pages 750–759, 1994.

[9] Marc Lanctot, Kevin Waugh, Martin Zinkevich, and Michael Bowling. Monte Carlo sampling for regret minimization in extensive games. In *Advances in Neural Information Processing Systems 22 (NIPS)*, pages 1078–1086, 2009.

[10] Marc Lanctot, Kevin Waugh, Martin Zinkevich, and Michael Bowling. Monte Carlo sampling for regret minimization in extensive games. Technical Report TR09-15, University of Alberta, 2009.

[11] Martin Zinkevich, Michael Johanson, Michael Bowling, and Carmelo Piccione. Regret minimization in games with incomplete information. Technical Report TR07-14, University of Alberta, 2007.

[12] Martin Zinkevich, Michael Johanson, Michael Bowling, and Carmelo Piccione. Regret minimization in games with incomplete information. In *Advances in Neural Information Processing Systems 20 (NIPS)*, pages 905–912, 2008.

